# Policy gradients in linearly-solvable MDPs

**Emanuel Todorov**
Applied Mathematics and Computer Science & Engineering
University of Washington
`todorov@cs.washington.edu`

## Abstract

We present policy gradient results within the framework of linearly-solvable MDPs. For the first time, compatible function approximators and natural policy gradients are obtained by estimating the cost-to-go function, rather than the (much larger) state-action advantage function as is necessary in traditional MDPs. We also develop the first compatible function approximators and natural policy gradients for continuous-time stochastic systems.

## 1  Introduction

Policy gradient methods [18] in Reinforcement Learning have gained popularity, due to the guaranteed improvement in control performance over iterations (which is often lacking in approximate policy or value iteration) as well as the discovery of more efficient gradient estimation methods. In particular it has been shown that one can replace the true advantage function with a compatible function approximator without affecting the gradient [8,14], and that a natural policy gradient (with respect to Fisher information) can be computed [2,5,11].

The goal of this paper is to apply policy gradient ideas to the linearly-solvable MDPs (or LMDPs) we have recently-developed [15, 16], as well as to a class of continuous stochastic systems with similar properties [4, 7, 16]. This framework has already produced a number of unique results – such as linear Bellman equations, general estimation-control dualities, compositionality of optimal control laws, path-integral methods for optimal control, etc. The present results with regard to policy gradients are also unique, as summarized in Abstract. While the contribution is mainly theoretical and scaling to large problems is left for future work, we provide simulations demonstrating rapid convergence. The paper is organized in two sections, treating discrete and continuous problems.

## 2  Discrete problems

Since a number of papers on LMDPs have already been published, we will not repeat the general development and motivation here, but instead only summarize the background needed for the present paper. We will then develop the new results regarding policy gradients.

### 2.1  Background on LMDPs

An LMDP is defined by a state cost $q(x)$ over a (discrete for now) state space $\mathcal{X}$, and a transition probability density $p(x'|x)$ corresponding to the notion of passive dynamics. In this paper we focus on infinite-horizon average-cost problems where $p(x'|x)$ is assumed to be ergodic, i.e. it has a unique stationary density. The admissible "actions" are all transition probability densities $\pi(x'|x)$ which are ergodic and satisfy $\pi(x'|x) = 0$ whenever $p(x'|x) = 0$. The cost function is

$$\ell(x, \pi(\cdot|x)) = q(x) + D_{\mathrm{KL}}(\pi(\cdot|x)\,||\,p(\cdot|x)) \tag{1}$$

Thus the controller is free to modify the default/passive dynamics in any way it wishes, but incurs a control cost related to the amount of modification.

The average cost $c$ and differential cost-to-go $v(x)$ for given $\pi(x'|x)$ satisfy the Bellman equation

$$c + v(x) = q(x) + \sum_{x'} \pi(x'|x) \left( \log \frac{\pi(x'|x)}{p(x'|x)} + v(x') \right) \tag{2}$$

where $v(x)$ is defined up to a constant. The optimal $c^*$ and $v^*(x)$ can be shown to satisfy

$$c^* + v^*(x) = q(x) - \log \sum_{x'} p(x'|x) \exp(-v^*(x')) \tag{3}$$

and the optimal $\pi^*(x'|x)$ can be found in closed form given $v^*(x)$:

$$\pi^*(x'|x) = \frac{p(x'|x) \exp(-v^*(x'))}{\sum_y p(y|x) \exp(-v^*(y))} \tag{4}$$

Exponentiating equation (3) makes it linear in $\exp(-v^*(x))$, although this will not be used here.

## 2.2 Policy gradient for a general parameterization

Consider a parameterization $\pi(x'|x, \mathbf{w})$ which is *valid* in the sense that it satisfies the above conditions and $\nabla_{\mathbf{w}} \pi \triangleq \partial \pi / \partial \mathbf{w}$ exists for all $\mathbf{w} \in \mathbb{R}^n$. Let $\mu(x, \mathbf{w})$ be the corresponding stationary density. We will also need the pair-wise density $\mu(x, x', \mathbf{w}) = \mu(x, \mathbf{w}) \pi(x'|x, \mathbf{w})$. To avoid notational clutter we will suppress the dependence on $\mathbf{w}$ in most of the paper; keep in mind that all quantities that depend on $\pi$ are functions of $\mathbf{w}$.

Our objective here is to compute $\nabla_{\mathbf{w}} c$. This is done by differentiating the Bellman equation (2) and following the template from [14]. The result (see Supplement) is given by

**Theorem 1.** *The LMDP policy gradient for any valid parameterization is*

$$\nabla_{\mathbf{w}} c = \sum_x \mu(x) \sum_{x'} \nabla_{\mathbf{w}} \pi(x'|x) \left( \log \frac{\pi(x'|x)}{p(x'|x)} + v(x') \right) \tag{5}$$

Let us now compare (5) to the policy gradient in traditional MDPs [14], which is

$$\nabla_{\mathbf{w}} c = \sum_x \mu(x) \sum_a \nabla_{\mathbf{w}} \pi(a|x) Q(x, a) \tag{6}$$

Here $\pi(a|x)$ is a stochastic policy over actions (parameterized by $\mathbf{w}$) and $Q(x, a)$ is the corresponding state-action cost-to-go. The general form of (5) and (6) is similar, however the term $\log(\pi/p) + v$ in (5) cannot be interpreted as a $Q$-function. Indeed it is not clear what a $Q$-function means in the LMDP setting. On the other hand, while in traditional MDPs one has to estimate $Q$ (or rather the advantage function) in order to compute the policy gradient, it will turn out that in LMDPs it is sufficient to estimate $v$.

## 2.3 A suitable policy parameterization

The relation (4) between the optimal policy $\pi^*$ and the optimal cost-to-go $v^*$ suggests parameterizing $\pi$ as a $p$-weighted Gibbs distribution. Since linear function approximators have proven very successful, we will use an energy function (for the Gibbs distribution) which is linear in $\mathbf{w}$:

$$\pi(x'|x, \mathbf{w}) \triangleq \frac{p(x'|x) \exp(-\mathbf{w}^\mathsf{T} \mathbf{f}(x'))}{\sum_y p(y|x) \exp(-\mathbf{w}^\mathsf{T} \mathbf{f}(y))} \tag{7}$$

Here $\mathbf{f}(x) \in \mathbb{R}^n$ is a vector of features. One can verify that (7) is a valid parameterization. We will also need the $\pi$-expectation operator

$$\Pi[f](x) \triangleq \sum_y \pi(y|x) f(y) \tag{8}$$

defined for both scalar and vector functions over $\mathcal{X}$. The general result (5) is now specialized as

**Theorem 2.** *The LMDP policy gradient for parameterization (7) is*

$$\nabla_{\mathbf{w}} c = \sum_{x, x'} \mu(x, x') (\Pi[\mathbf{f}](x) - \mathbf{f}(x')) (v(x') - \mathbf{w}^\mathsf{T} \mathbf{f}(x')) \tag{9}$$

As expected from (4), we see that the energy function $\mathbf{w}^\mathsf{T} \mathbf{f}(x)$ and the cost-to-go $v(x)$ are related. Indeed if they are equal the gradient vanishes (the converse is not true).

## 2.4 Compatible cost-to-go function approximation

One of the more remarkable aspects of policy gradient results [8, 14] in traditional MDPs is that, when the true $Q$ function is replaced with a *compatible* approximation satisfying certain conditions, the gradient remains unchanged. Key to obtaining such results is making sure that the approximation error is orthogonal to the remaining terms in the expression for the policy gradient. Our goal in this section is to construct a compatible function approximator for LMDPs. The procedure is somewhat elaborate and unusual, so we provide the derivation before stating the result in Theorem 3 below.

Given the form of (9), it makes sense to approximate $v(x)$ as a linear combination of the same features $\mathbf{f}(x)$ used to represent the energy function: $\widehat{v}(x, \mathbf{r}) \triangleq \mathbf{r}^\mathsf{T} \mathbf{f}(x)$. Let us also define the approximation error $\varepsilon_\mathbf{r}(x) \triangleq v(x) - \widehat{v}(x, \mathbf{r})$. If the policy gradient $\nabla_\mathbf{w} c$ is to remain unchanged when $v$ is replaced with $\widehat{v}$ in (9), the following quantity must be zero:

$$\mathbf{d}(\mathbf{r}) \triangleq \sum_{x,x'} \mu(x, x') \left(\Pi[\mathbf{f}](x) - \mathbf{f}(x')\right) \varepsilon_\mathbf{r}(x') \tag{10}$$

Expanding (10) and using the stationarity of $\mu$, we can simplify $\mathbf{d}$ as

$$\mathbf{d}(\mathbf{r}) = \sum_x \mu(x) \left(\Pi[\mathbf{f}](x) \Pi[\varepsilon_\mathbf{r}](x) - \mathbf{f}(x) \varepsilon_\mathbf{r}(x)\right) \tag{11}$$

One can also incorporate an $x$-dependent baseline in (9), such as $v(x)$ which is often used in traditional MDPs. However the baseline vanishes after the simplification, and the result is again (11).

Now we encounter a complication. Suppose we were to fit $\widehat{v}$ to $v$ in a least-squares sense, i.e. minimize the squared error weighted by $\mu$. Denote the resulting weight vector $\mathbf{r}_{LS}$:

$$\mathbf{r}_{LS} \triangleq \arg\min_\mathbf{r} \sum_x \mu(x) \left(v(x) - \mathbf{r}^\mathsf{T} \mathbf{f}(x)\right)^2 \tag{12}$$

This is arguably the best fit one can hope for. The error $\varepsilon_\mathbf{r}$ is now orthogonal to the features $\mathbf{f}$, thus for $\mathbf{r} = \mathbf{r}_{LS}$ the second term in (11) vanishes, but the first term does not. Indeed we have verified numerically (on randomly-generated LMDPs) that $\mathbf{d}(\mathbf{r}_{LS}) \neq 0$.

If the best fit is not good enough, what are we to do? Recall that we do not actually need a good fit, but rather a vector $\mathbf{r}$ such that $\mathbf{d}(\mathbf{r}) = 0$. Since $\mathbf{d}(\mathbf{r})$ and $\mathbf{r}$ are linearly related and have the same dimensionality, we can directly solve this equation for $\mathbf{r}$. Replacing $\varepsilon_\mathbf{r}(x)$ with $v(x) - \mathbf{r}^\mathsf{T} \mathbf{f}(x)$ and using the fact that $\Pi$ is a linear operator, we have $\mathbf{d}(\mathbf{r}) = A\mathbf{r} - \mathbf{k}$ where

$$A \triangleq \sum_x \mu(x) \left(\mathbf{f}(x) \mathbf{f}(x)^\mathsf{T} - \Pi[\mathbf{f}](x) \Pi[\mathbf{f}](x)^\mathsf{T}\right) \tag{13}$$

$$\mathbf{k} \triangleq \sum_x \mu(x) \left(\mathbf{f}(x) v(x) - \Pi[\mathbf{f}](x) \Pi[v](x)\right)$$

We are not done yet because $\mathbf{k}$ still depends on $v$. The goal now is to approximate $v$ in such a way that $\mathbf{k}$ remains unchanged. To this end we use (2) and express $\Pi[v]$ in terms of $v$:

$$c + v(x) - \ell(x) = \Pi[v](x) \tag{14}$$

Here $\ell(x)$ is shortcut notation for $\ell(x, \pi(\cdot|x, \mathbf{w}))$. Thus the vector $\mathbf{k}$ becomes

$$\mathbf{k} = \sum_x \mu(x) \left(\mathbf{g}(x) v(x) + \Pi[\mathbf{f}](x) (\ell(x) - c)\right) \tag{15}$$

where the policy-specific auxiliary features $\mathbf{g}(x)$ are related to the original features $\mathbf{f}(x)$ as

$$\mathbf{g}(x) \triangleq \mathbf{f}(x) - \Pi[\mathbf{f}](x) \tag{16}$$

The second term in (15) does not depend on $v$; it only depends on $c = \sum_x \mu(x) \ell(x)$. The first term in (15) involves the projection of $v$ on the auxiliary features $\mathbf{g}$. This projection can be computed by defining the auxiliary function approximator $\widetilde{v}(x, \mathbf{s}) \triangleq \mathbf{s}^\mathsf{T} \mathbf{g}(x)$ and fitting it to $v$ in a least-squares sense, as in (12) but using $\mathbf{g}(x)$ rather than $\mathbf{f}(x)$. The approximation error is now orthogonal to the auxiliary features $\mathbf{g}(x)$, and so replacing $v(x)$ with $\widetilde{v}(x, \mathbf{s})$ in (15) does not affect $\mathbf{k}$. Thus we have

**Theorem 3.** *The following procedure yields the exact LMDP policy gradient:*

1. fit $\widetilde{v}(x, \mathbf{s})$ to $v(x)$ in a least squares sense, and also compute $c$
2. compute $A$ from (13), and $\mathbf{k}$ from (15) by replacing $v(x)$ with $\widetilde{v}(x, \mathbf{s})$
3. "fit" $\widehat{v}(x, \mathbf{r})$ by solving $A\mathbf{r} = \mathbf{k}$
4. the policy gradient is

$$\nabla_\mathbf{w} c = \sum_{x,x'} \mu(x, x') \left(\mathbf{f}(x') - \Pi[\mathbf{f}](x)\right) \mathbf{f}(x')^\mathsf{T} (\mathbf{w} - \mathbf{r}) \tag{17}$$

This is the first policy gradient result with compatible function approximation over the state space rather than the state-action space. The computations involve averaging over $\mu$, which in practice will be done through sampling (see below). The requirement that $v - \widetilde{v}$ be orthogonal to $\mathbf{g}$ is somewhat restrictive, however an equivalent requirement arises in traditional MDPs [14].

## 2.5 Natural policy gradient

When the parameter space has a natural metric $G(\mathbf{w})$, optimization algorithms tend to work better if the gradient of the objective function is pre-multiplied by $G(\mathbf{w})^{-1}$. This yields the so-called natural gradient [1]. In the context of policy gradient methods [5, 11] where $\mathbf{w}$ parameterizes a probability density, the natural metric is given by Fisher information (which depends on $x$ because $\mathbf{w}$ parameterizes the conditional density). Averaging over $\mu$ yields the metric

$$G(\mathbf{w}) \triangleq \sum_{x,x'} \mu(x,x') \nabla_{\mathbf{w}} \log \pi(x'|x) \nabla_{\mathbf{w}} \log \pi(x'|x)^{\mathsf{T}} \qquad (18)$$

We then have the following result (see Supplement):

**Theorem 4**. *With the vector $\mathbf{r}$ computed as in Theorem 3, the LMDP natural policy gradient is*

$$G(\mathbf{w})^{-1} \nabla_{\mathbf{w}} c = \mathbf{w} - \mathbf{r} \qquad (19)$$

Let us compare this result to the natural gradient in traditional MDPs [11], which is

$$G(\mathbf{w})^{-1} \nabla_{\mathbf{w}} c = \mathbf{r} \qquad (20)$$

In traditional MDPs one maximizes reward while in LMDPs one minimizes cost, thus the sign difference. Recall that in traditional MDPs the policy $\pi$ is parameterized using features over the state-action space while in LMDPs we only need features over the state space. Thus the vectors $\mathbf{w}, \mathbf{r}$ will usually have lower dimensionality in (19) compared to (20).

Another difference is that in LMDPs the (regular as well as natural) policy gradient vanishes when $\mathbf{w} = \mathbf{r}$, which is a sensible fixed-point condition. In traditional MDPs the policy gradient vanishes when $\mathbf{r} = 0$, which is peculiar because it corresponds to the advantage function approximation being identically $0$. The true advantage function is of course different, but if the policy becomes deterministic and only one action is sampled per state, the resulting data can be fit with $\mathbf{r} = 0$. Thus any deterministic policy is a local maximum in traditional MDPs. At these local maxima the policy gradient theorem cannot actually be applied because it requires a stochastic policy. When the policy becomes near-deterministic, the number of samples needed to obtain accurate estimates increases because of the lack of exploration [6]. These issues do not seem to arise in LMDPs.

## 2.6 A Gauss-Newton method for approximating the optimal cost-to-go

Instead of using policy gradient, we can solve (3) for the optimal $v^*$ directly. One option is approximate policy iteration – which in our context takes on a simple form. Given the policy parameters $\mathbf{w}^{(i)}$ at iteration $i$, approximate the cost-to-go function and obtain the feature weights $\mathbf{r}^{(i)}$, and then set $\mathbf{w}^{(i+1)} = \mathbf{r}^{(i)}$. This is equivalent to the above natural gradient method with step size 1, using a biased approximator instead of the compatible approximator given by Theorem 3.

The other option is approximate value iteration – which is a fixed-point method for solving (3) while replacing $v^*(x)$ with $\mathbf{w}^{\mathsf{T}}\mathbf{f}(x)$. We can actually do better than value iteration here. Since (3) has already been optimized over the controls and is differentiable, we can apply an efficient Gauss-Newton method. Up to an additive constant $c$, the Bellman error from (3) is

$$e(x, \mathbf{w}) \triangleq \mathbf{w}^{\mathsf{T}}\mathbf{f}(x) - q(x) + \log \sum_y p(y|x) \exp\left(-\mathbf{w}^{\mathsf{T}}\mathbf{f}(y)\right) \qquad (21)$$

Interestingly, the gradient of this Bellman error coincides with our auxilliary features $\mathbf{g}$:

$$\nabla_{\mathbf{w}} e(x, \mathbf{w}) = \mathbf{f}(x) - \sum_y \frac{p(y|x) \exp\left(-\mathbf{w}^{\mathsf{T}}\mathbf{f}(y)\right)}{\sum_s p(s|x) \exp\left(-\mathbf{w}^{\mathsf{T}}\mathbf{f}(s)\right)} \mathbf{f}(y) = \mathbf{f}(x) - \Pi[\mathbf{f}](x) = \mathbf{g}(x) \qquad (22)$$

where $\Pi$ and $\mathbf{g}$ are the same as in (16, 8). We now linearize: $e(x, \mathbf{w} + \delta\mathbf{w}) \approx e(x, \mathbf{w}) + \delta\mathbf{w}^{\mathsf{T}}\mathbf{g}(x)$ and proceed to minimize (with respect to $c$ and $\delta\mathbf{w}$) the quantity

$$\sum_x \bar{\mu}(x) \left(c + e(x, \mathbf{w}) + \delta\mathbf{w}^{\mathsf{T}}\mathbf{g}(x)\right)^2 \qquad (23)$$

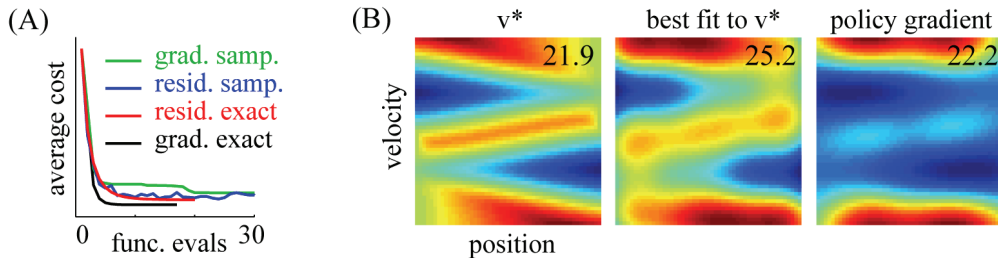

Figure 1: (A) Learning curves for a random LMDP. "resid" is the Gauss-Newton method. The sampling versions use 400 samples per evaluation: 20 trajectories with 20 steps each, starting from the stationary distribution. (B) Cost-to-go functions for the metronome LMDP. The numbers show the average costs obtained. There are 2601 discrete states and 25 features (Gaussians). Convergence was observed in about 10 evaluations (of the objective and the gradient) for both algorithms, exact and sampling versions. The sampling version of the Gauss-Newton method worked well with 400 samples per evaluation; the natural gradient needed around 2500 samples.

Normally the density $\overline{\mu}(x)$ would be fixed, however we have found empirically that the resulting algorithm yields better policies if we set $\overline{\mu}(x)$ to the policy-specific stationary density $\mu(x, \mathbf{w})$ at each iteration. It is not clear how to guarantee convergence of this algorithm given that the objective function itself is changing over iterations, but in practice we observed that simple damping is sufficient to make it convergent (e.g. $\mathbf{w} \leftarrow \mathbf{w} + \delta\mathbf{w}/2$).

It is notable that minimization of (23) is closely related to policy evaluation via Bellman residual minimization. More precisely, using (14, 16) it is easy to see that TD(0) applied to our problem would seek to minimize

$$\sum_x \mu(x, \mathbf{w}) \left( c - \ell(x, \mathbf{w}) + \mathbf{r}^\mathsf{T}\mathbf{g}(x) \right)^2 \tag{24}$$

The similarity becomes even more apparent if we write $-\ell(x, \mathbf{w})$ more explicitly as

$$-\ell(x, \mathbf{w}) = \mathbf{w}^\mathsf{T}\Pi\left[\mathbf{f}\right](x) - q(x) + \log\sum_y p(y|x)\exp\left(-\mathbf{w}^\mathsf{T}\mathbf{f}(y)\right) \tag{25}$$

Thus the only difference from (21) is that one expression has the term $\mathbf{w}^\mathsf{T}\mathbf{f}(x)$ at the place where the other expression has the term $\mathbf{w}^\mathsf{T}\Pi\left[\mathbf{f}\right](x)$. Note that the Gauss-Newton method proposed here would be expected to have second-order convergence, even though the amount of computation/sampling per iteration is the same as in a policy gradient method.

### 2.7 Numerical experiments

We compared the natural policy gradient and the Gauss-Newton method, both in exact form and with sampling, on two classes of LMDPs: randomly generated, and a discretization of a continuous "metronome" problem taken from [17]. Fitting the auxiliary approximator $\widetilde{v}(x, \mathbf{s})$ was done using the LSTD($\lambda$) algorithm [3]. Note that Theorem 3 guarantees compatibility only for $\lambda = 1$, however lower values of $\lambda$ reduce variance and still provide good descent directions in practice (as one would expect). We ended up using $\lambda = 0.2$ after some experimentation. The natural gradient was used with the BFGS minimizer "minFunc" [12].

Figure 1A shows typical learning curves on a random LMDP with 100 states, 20 random features, and random passive dynamics with $50\%$ sparsity. In this case the algorithms had very similar performance. On other examples we observed one or the other algorithm being slightly faster or producing better minima, but overall they were comparable. The average cost of the policies found by the Gauss-Newton method occasionally increased towards the end of the iteration.

Figure 1B compares the optimal cost-to-go $v^*$, the least-squares fit to the known $v^*$ using our features (which were a 5-by-5 grid of Gaussians), and the solution of the policy gradient method initialized with $\mathbf{w} = 0$. Note that the latter has lower cost compared to the least-squares fit. In this case both algorithms converged in about 10 iterations, although the Gauss-Newton method needed about 5 times fewer samples in order to achieve similar performance to the exact version.

# 3 Continuous problems

Unlike the discrete case where we focused exclusively on LMDPs, here we begin with a very general problem formulation and present interesting new results. These results are then specialized to a narrower class of problems which are continuous (in space and time) but nevertheless have similar properties to LMDPs.

## 3.1 Policy gradient for general controlled diffusions

Consider the controlled Ito diffusion

$$d\mathbf{x} = \mathbf{b}(\mathbf{x}, \mathbf{u})\,dt + C(\mathbf{x})\,d\omega \tag{26}$$

where $\omega(t)$ is a standard multidimensional Brownian motion process, and $\mathbf{u}$ is now a traditional control vector. Let $\ell(\mathbf{x}, \mathbf{u})$ be a cost function. As before we focus on infinite-horizon average-cost optimal control problems. Given a policy $\mathbf{u} = \pi(\mathbf{x})$, the average cost $c$ and differential cost-to-go $v(\mathbf{x})$ satisfy the Hamilton-Jacobi-Bellman (HJB) equation

$$c = \ell(\mathbf{x}, \pi(\mathbf{x})) + \mathcal{L}[v](\mathbf{x}) \tag{27}$$

where $\mathcal{L}$ is the following 2nd-order linear differential operator:

$$\mathcal{L}[v](\mathbf{x}) \triangleq \mathbf{b}(\mathbf{x}, \pi(\mathbf{x}))^{\mathsf{T}} \nabla_{\mathbf{x}} v(\mathbf{x}) + \tfrac{1}{2} \operatorname{trace}\left( C(\mathbf{x}) C(\mathbf{x})^{\mathsf{T}} \nabla_{\mathbf{xx}} v(\mathbf{x}) \right) \tag{28}$$

In can be shown [10] that $\mathcal{L}$ coincides with the infinitesimal generator of (26), i.e. it computes the expected directional derivative of $v$ along trajectories generated by (26). We will need

**Lemma 1**. *Let $\mathcal{L}$ be the infinitesimal generator of an Ito diffusion which has a stationary density $\mu$, and let $f$ be a twice-differentiable function. Then*

$$\int \mu(\mathbf{x})\,\mathcal{L}[f](\mathbf{x})\,d\mathbf{x} = 0 \tag{29}$$

**Proof:** The adjoint $\mathcal{L}^*$ of the infinitesimal generator $\mathcal{L}$ is known to be the Fokker-Planck operator – which computes the time-evolution of a density under the diffusion [10]. Since $\mu$ is the stationary density, $\mathcal{L}^*[\mu](\mathbf{x}) = 0$ for all $\mathbf{x}$, and so $\langle \mathcal{L}^*[\mu], f \rangle = 0$. Since $\mathcal{L}$ and $\mathcal{L}^*$ are adjoint, $\langle \mathcal{L}^*[\mu], f \rangle = \langle \mu, \mathcal{L}[f] \rangle$. Thus $\langle \mu, \mathcal{L}[f] \rangle = 0$.

This lemma seems important-yet-obvious so we would not be surprised if it was already known, but we have not seen in the literature. Note that many diffusions lack stationary densities. For example the density of Brownian motion initialized at the origin is a zero-mean Gaussian whose covariance grows linearly with time – thus there is no stationary density. If however the diffusion is controlled and the policy tends to keep the state within some region, then a stationary density would normally exist. The existence of a stationary density may actually be a sensible definition of stability for stochastic systems (although this point will not be pursued in the present paper).

Now consider any policy parameterization $\mathbf{u} = \pi(\mathbf{x}, \mathbf{w})$ such that (for the current value of $\mathbf{w}$) the diffusion (26) has a stationary density $\mu$ and $\nabla_{\mathbf{w}}\pi$ exists. Differentiating (27), and using the shortcut notation $\mathbf{b}(\mathbf{x})$ in place of $\mathbf{b}(\mathbf{x}, \pi(\mathbf{x}, \mathbf{w}))$ and similarly for $\ell(\mathbf{x})$, we have

$$\nabla_{\mathbf{w}} c = \nabla_{\mathbf{w}} \ell(\mathbf{x}) + \nabla_{\mathbf{w}} \mathbf{b}(\mathbf{x})^{T} \nabla_{\mathbf{x}} v(\mathbf{x}) + \mathcal{L}[\nabla_{\mathbf{w}} v](\mathbf{x}) \tag{30}$$

Here $\mathcal{L}[\nabla_{\mathbf{w}} v]$ is meant component-wise. If we now average over $\mu$, the last term will vanish due to Lemma 1. This is essential for a policy gradient procedure which seeks to avoid finite differencing; indeed $\nabla_{\mathbf{w}} v$ could not be estimated while sampling from a single policy. Thus we have

**Theorem 5**. *The policy gradient of the controlled diffusion (26) is*

$$\nabla_{\mathbf{w}} c = \int \mu(\mathbf{x}) \left( \nabla_{\mathbf{w}} \ell(\mathbf{x}) + \nabla_{\mathbf{w}} \mathbf{b}(\mathbf{x})^{T} \nabla_{\mathbf{x}} v(\mathbf{x}) \right) d\mathbf{x} \tag{31}$$

Unlike most other results in stochastic optimal control, equation (31) does not involve the Hessian $\nabla_{\mathbf{xx}} v$, although we can obtain a $\nabla_{\mathbf{xx}} v$-dependent term here if we allow $C$ to depend on $\mathbf{u}$. We now illustrate Theorem 5 on a linear-quadratic-Gaussian (LQG) control problem.

**Example (LQG).** Consider dynamics $dx = udt + d\omega$ and cost $\ell(x, u) = x^2 + u^2$. Let $u = -wx$ be the parameterized policy with $w > 0$. The differential cost-to-go is known to be in the form $v(x) = sx^2$. Substituting in the HJB equation and matching powers of $x$ yields $c = s = \frac{w^2+1}{2w}$, and so the policy gradient can be computed directly as $\nabla_w c = 1 - \frac{w^2+1}{2w^2}$. The stationary density $\mu(x)$ is a zero-mean Gaussian with variance $\sigma^2 = \frac{1}{2w}$. One can now verify that the gradient given by Theorem 5 is identical to the $\nabla_w c$ computed above.

Another interesting aspect of Theorem 5 is that it is a natural generalization of classic results from finite-horizon deterministic optimal control [13], even though it cannot be derived from those results. Suppose we have an open-loop control trajectory $\mathbf{u}(t), 0 \le t \le T$, the resulting state trajectory (starting from a given $\mathbf{x}_0$) is $\mathbf{x}(t)$, and the corresponding co-state trajectory (obtained by integrating Pontryagin's ODE backwards in time) is $\lambda(t)$. It is known that the gradient of the total cost $J$ w.r.t. $\mathbf{u}$ is $\nabla_{\mathbf{u}}\ell + \nabla_{\mathbf{u}}\mathbf{b}^\mathsf{T}\lambda$. Now suppose $\mathbf{u}(t)$ is parameterized by some vector $\mathbf{w}$. Then

$$\nabla_\mathbf{w} J = \int \nabla_\mathbf{w} \mathbf{u}(t)^\mathsf{T} \nabla_{\mathbf{u}(t)} J dt = \int \left( \nabla_\mathbf{w}\ell(\mathbf{x}(t), \mathbf{u}(t)) + \nabla_\mathbf{w}\mathbf{b}(\mathbf{x}(t), \mathbf{u}(t))^\mathsf{T}\lambda(t) \right) dt \quad (32)$$

The co-state $\lambda(t)$ is known to be equal to the gradient $\nabla_\mathbf{x} v(\mathbf{x}, t)$ of the cost-to-go function for the (closed-loop) deterministic problem. Thus (31) and (32) are very similar. Of course in finite-horizon settings there is no stationary density, and instead the integral in (32) is over the trajectory. An RL method for estimating $\nabla_\mathbf{w} J$ in deterministic problems was developed in [9].

Theorem 5 suggests a simple procedure for estimating the policy gradient via sampling: fit a function approximator $\widehat{v}$ to $v$, and use $\nabla_\mathbf{x}\widehat{v}$ in (31). Alternatively, a compatible approximation scheme can be obtained by fitting $\nabla_\mathbf{x}\widehat{v}$ to $\nabla_\mathbf{x} v$ in a least-squares sense, using a linear approximator with features $\nabla_\mathbf{w}\mathbf{b}(\mathbf{x})$. This however is not practical because learning targets for $\nabla_\mathbf{x} v$ are difficult to obtain. Ideally we would construct a compatible approximation scheme which involves fitting $\widehat{v}$ rather than $\nabla_\mathbf{x}\widehat{v}$. It is not clear how to do that for general diffusions, but can be done for a restricted problem class as shown next.

## 3.2 Natural gradient and compatible approximation for linearly-solvable diffusions

We now focus on a more restricted family of stochastic optimal control problems which arise in many situations (e.g. most mechanical systems can be described in this form):

$$d\mathbf{x} = (\mathbf{a}(\mathbf{x}) + B(\mathbf{x})\mathbf{u})\, dt + C(\mathbf{x})\, d\omega \quad (33)$$

$$\ell(\mathbf{x}, \mathbf{u}) = q(x) + \tfrac{1}{2}\mathbf{u}^\mathsf{T} R(\mathbf{x})\mathbf{u}$$

Such problems have been studied extensively [13]. The optimal control law $\mathbf{u}^*$ and the optimal differential cost-go-to $v^*(\mathbf{x})$ are known to be related as $\mathbf{u}^* = -R^{-1}B^\mathsf{T}\nabla_\mathbf{x} v^*$. As in the discrete case we use this relation to motivate the choice of policy parameterization and cost-to-go function approximator. Choosing some features $\mathbf{f}(\mathbf{x})$, we define $\widehat{v}(\mathbf{x}, \mathbf{r}) \triangleq \mathbf{r}^\mathsf{T}\mathbf{f}(\mathbf{x})$ as before, and

$$\pi(\mathbf{x}, \mathbf{w}) \triangleq -R(\mathbf{x})^{-1} B(\mathbf{x})^\mathsf{T} \nabla_\mathbf{x}\left(\mathbf{w}^\mathsf{T}\mathbf{f}(\mathbf{x})\right) \quad (34)$$

It is convenient to also define the matrix $F(\mathbf{x}) \triangleq \nabla_\mathbf{x}\mathbf{f}(\mathbf{x})^\mathsf{T}$, so that $\nabla_\mathbf{x}\widehat{v}(\mathbf{x}, \mathbf{r}) = F(\mathbf{x})\mathbf{r}$. We can now substitute these definitions in the general result (31), replace $v$ with the approximation $\widehat{v}$, and skipping the algebra, obtain the corresponding approximation to the policy gradient:

$$\widetilde{\nabla}_\mathbf{w} c = \int \mu(\mathbf{x}) F(\mathbf{x})^\mathsf{T} B(\mathbf{x}) R(\mathbf{x})^{-1} B(\mathbf{x})^\mathsf{T} F(\mathbf{x}) (\mathbf{w} - \mathbf{r})\, d\mathbf{x} \quad (35)$$

Before addressing the issue of compatibility (i.e. whether $\widetilde{\nabla}_\mathbf{w} c = \nabla_\mathbf{w} c$), we seek a natural gradient version of (35). To this end we need to interpret $F^\mathsf{T} BR^{-1}B^\mathsf{T} F$ as Fisher information for the (infinitesimal) transition probability density of our parameterized diffusion. We do this by discretizing the time axis with time step $h$, and then dividing by $h$. The $h$-step explicit Euler discretization of the stochastic dynamics (33) is given by the Gaussian

$$p_h(\cdot | \mathbf{x}, \mathbf{w}) = \mathcal{N}\left(\mathbf{x} + h\mathbf{a}(\mathbf{x}) - hB(\mathbf{x}) R(\mathbf{x})^{-1} B(\mathbf{x})^\mathsf{T} F(\mathbf{x})\mathbf{w};\; hC(\mathbf{x}) C(\mathbf{x})^\mathsf{T}\right) \quad (36)$$

Suppressing the dependence on $\mathbf{x}$, Fisher information becomes

$$\frac{1}{h}\int p_h \nabla_\mathbf{w} \log p_h \nabla_\mathbf{w} \log p_h^\mathsf{T} d\mathbf{x}' = F^\mathsf{T} BR^{-1}B^\mathsf{T} \left(CC^\mathsf{T}\right)^{-1} BR^{-1}B^\mathsf{T} F \quad (37)$$

Comparing to (35) we see that a natural gradient result is obtained when

$$C\left(\mathbf{x}\right)C\left(\mathbf{x}\right)^{\mathsf{T}} = B\left(\mathbf{x}\right)R\left(\mathbf{x}\right)^{-1}B\left(\mathbf{x}\right)^{\mathsf{T}} \tag{38}$$

Assuming (38) is satisfied, and defining $G\left(\mathbf{w}\right)$ as the average of Fisher information over $\mu\left(\mathbf{x}\right)$,

$$G\left(\mathbf{w}\right)^{-1}\widetilde{\nabla}_{\mathbf{w}}c = \mathbf{w} - \mathbf{r} \tag{39}$$

Condition (38) is rather interesting. Elsewhere we have shown [16] that the same condition is needed to make problem (33) linearly-solvable. More precisely, the exponentiated HJB equation for the optimal $v^*$ in problem (33, 38) is linear in $\exp\left(-v^*\right)$. We have also shown [16] that the continuous problem (33, 38) is the limit (when $h \to 0$) of continuous-state discrete-time LMDPs constructed via Euler discretization as above. The compatible function approximation scheme from Theorem 3 can then be applied to these LMDPs. Recall (8). Since $\mathcal{L}$ is the infinitesimal generator, for any twice-differentiable function $f$ we have

$$\Pi\left[f\right]\left(\mathbf{x}\right) = f\left(\mathbf{x}\right) + h\mathcal{L}\left[f\right]\left(\mathbf{x}\right) + o\left(h^2\right) \tag{40}$$

Substituting in (13), dividing by $h$ and taking the limit $h \to 0$, the matrix $A$ and vector $\mathbf{k}$ become

$$A = \int \mu\left(\mathbf{x}\right)\left(-\mathcal{L}\left[\mathbf{f}\right]\left(\mathbf{x}\right)\mathbf{f}\left(\mathbf{x}\right)^{\mathsf{T}} - \mathbf{f}\left(\mathbf{x}\right)\mathcal{L}\left[\mathbf{f}\right]\left(\mathbf{x}\right)^{\mathsf{T}}\right)d\mathbf{x} \tag{41}$$

$$\mathbf{k} = \int \mu\left(\mathbf{x}\right)\left(-\mathcal{L}\left[\mathbf{f}\right]\left(\mathbf{x}\right)v\left(\mathbf{x}\right) + \mathbf{f}\left(\mathbf{x}\right)\left(\ell\left(\mathbf{x}\right) - c\right)\right)d\mathbf{x}$$

Compatibility is therefore achieved when the approximation error in $v$ is orthogonal to $\mathcal{L}\left[\mathbf{f}\right]$. Thus the auxiliary function approximator is now $\widetilde{v}\left(\mathbf{x},\mathbf{s}\right) \triangleq \mathbf{s}^{\mathsf{T}}\mathcal{L}\left[\mathbf{f}\right]\left(\mathbf{x}\right)$, and we have

**Theorem 6.** *The following procedure yields the exact policy gradient for problem (33, 38):*

1. fit $\widetilde{v}\left(\mathbf{x},\mathbf{s}\right)$ to $v\left(\mathbf{x}\right)$ in a least-squares sense, and also compute $c$
2. compute $A$ and $\mathbf{k}$ from (41), replacing $v\left(\mathbf{x}\right)$ with $\widetilde{v}\left(\mathbf{x},\mathbf{s}\right)$
3. "fit" $\widehat{v}\left(\mathbf{x},\mathbf{r}\right)$ by solving $A\mathbf{r} = \mathbf{k}$
4. the policy gradient is (35), and the natural policy gradient is (39)

This is the first policy gradient result with compatible function approximation for continuous stochastic systems. It is very similar to the corresponding results in the discrete case (Theorems 3,4) except it involves the differential operator $\mathcal{L}$ rather than the integral operator $\Pi$.

## 4   Summary

Here we developed compatible function approximators and natural policy gradients which only require estimation of the cost-to-go function. This was possible due to the unique properties of the LMDP framework. The resulting approximation scheme is unusual, using policy-specific auxiliary features derived from the primary features. In continuous time we also obtained a new policy gradient result for control problems that are not linearly-solvable, and showed that it generalizes results from deterministic optimal control. We also derived a somewhat heuristic but nevertheless promising Gauss-Newton method for solving for the optimal cost-to-go directly; it appears to be a hybrid between value iteration and policy gradient.

One might wonder why we need policy gradients here given that the (exponentiated) Bellman equation is linear, and approximating its solution using features is faster than any other procedure in Reinforcement Learning and Approximate Dynamic Programming. The answer is that minimizing Bellman error does not always give the best policy – as illustrated in Figure 1B. Indeed a combined approach may be optimal: solve the linear Bellman equation approximately [17], and then use the solution to initialize the policy gradient method. This idea will be explored in future work.

Our new methods require a model – as do all RL methods that rely on state values rather than state-action values. We do not see this as a shortcoming because, despite all the effort that has gone into model-free RL, the resulting methods do not seem applicable to truly complex optimal control problems. Our methods involve model-based sampling which combines the best of both worlds: computational speed, and grounding in reality (assuming we have a good model of reality).

**Acknowledgements**.

This work was supported by the US National Science Foundation. Thanks to Guillaume Lajoie and Jan Peters for helpful discussions.

# References

[1] S. Amari. Natural gradient works efficiently in learning. *Neural Computation*, 10:251–276, 1998.

[2] J. Bagnell and J. Schneider. Covariant policy search. In *International Joint Conference on Artificial Intelligence*, 2003.

[3] J. Boyan. Least-squares temporal difference learning. In *International Conference on Machine Learning*, 1999.

[4] W. Fleming and S. Mitter. Optimal control and nonlinear filtering for nondegenerate diffusion processes. *Stochastics*, 8:226–261, 1982.

[5] S. Kakade. A natural policy gradient. In *Advances in Neural Information Processing Systems*, 2002.

[6] S. Kakade. *On the Sample Complexity of Reinforcement Learning*. PhD thesis, University College London, 2003.

[7] H. Kappen. Linear theory for control of nonlinear stochastic systems. *Physical Review Letters*, 95, 2005.

[8] V. Konda and J. Tsitsiklis. Actor-critic algorithms. *SIAM Journal on Control and Optimization*, pages 1008–1014, 2001.

[9] R. Munos. Policy gradient in continuous time. *The Journal of Machine Learning Research*, 7:771–791, 2006.

[10] B. Oksendal. *Stochastic Differential Equations (4th Ed)*. Springer-Verlag, Berlin, 1995.

[11] J. Peters and S. Schaal. Natural actor-critic. *Neurocomputing*, 71:1180–1190, 2008.

[12] M. Schmidt. minfunc. online material, 2005.

[13] R. Stengel. *Optimal Control and Estimation*. Dover, New York, 1994.

[14] R. Sutton, D. Mcallester, S. Singh, and Y. Mansour. Policy gradient methods for reinforcement learning with function approximation. In *Advances in Neural Information Processing Systems*, 2000.

[15] E. Todorov. Linearly-solvable Markov decision problems. *Advances in Neural Information Processing Systems*, 2006.

[16] E. Todorov. Efficient computation of optimal actions. *PNAS*, 106:11478–11483, 2009.

[17] E. Todorov. Eigen-function approximation methods for linearly-solvable optimal control problems. *IEEE ADPRL*, 2009.

[18] R. Williams. Simple statistical gradient following algorithms for connectionist reinforcement learning. *Machine Learning*, pages 229–256, 1992.

